# Newscast EM

**Wojtek Kowalczyk**
Department of Computer Science
Vrije Universiteit Amsterdam
The Netherlands
wojtek@cs.vu.nl

**Nikos Vlassis**
Informatics Institute
University of Amsterdam
The Netherlands
vlassis@science.uva.nl

## Abstract

We propose a gossip-based distributed algorithm for Gaussian mixture learning, Newscast EM. The algorithm operates on network topologies where each node observes a local quantity and can communicate with other nodes in an arbitrary point-to-point fashion. The main difference between Newscast EM and the standard EM algorithm is that the M-step in our case is implemented in a decentralized manner: (random) pairs of nodes repeatedly exchange their local parameter estimates and combine them by (weighted) averaging. We provide theoretical evidence and demonstrate experimentally that, under this protocol, nodes converge exponentially fast to the correct estimates in each M-step of the EM algorithm.

## 1 Introduction

Advances in network technology, like peer-to-peer networks on the Internet or sensor networks, have highlighted the need for efficient ways to deal with large amounts of data that are distributed over a set of nodes. Examples are financial data reported on the Internet, weather data observed by a set of sensors, etc. In particular, in many data mining applications we are interested in learning a *global* model from such data, like a probability distribution or a clustering of the data, without first transferring all the data to a central repository. Ideally, we would like to have a fully decentralized algorithm that computes and disseminates *aggregates* of the data, with minimal processing and communication requirements and good fault-tolerant behavior.

A recent development in distributed systems technology is the use of *gossip-based* models of computation [1, 2, 3]. Roughly, in a gossip-based protocol each node repeatedly contacts some other node at random and the two nodes exchange information. Gossip-based protocols are very simple to implement, while they enjoy strong performance guarantees as a result of randomization. Their use in data mining and machine learning applications is currently finding inroads [4, 5].

In this paper we propose a gossip-based, fully decentralized implementation of the Expectation-Maximization (EM) algorithm for Gaussian mixture learning [6]. Our algorithm, which we call 'Newscast EM', assumes a set of data $\{x_i\}$ that are drawn independently from a common Gaussian mixture and are distributed over the nodes of a network (one data point per node). Newscast EM utilizes a gossip-based protocol in its M-step to

learn a global Gaussian mixture model $p(x)$ from the data. The main idea is to perform the M-step in a number of cycles. Each node starts with a local estimate of the model parameters. Then, in every cycle, each node contacts some other node that is chosen at random from a list of known nodes, and the two nodes replace their local model estimates by their (weighted) averages. As we show below, under such a protocol the (erroneous) local models of the individual nodes converge exponentially fast to the (correct) global model in each M-step of the algorithm.

Our approach is fundamentally different from other distributed exact implementations of the EM algorithm that resort on global broadcasting [7] or routing trees [8]. In the latter, for instance, data sufficient statistics are propagated through a spanning tree in the network, combined with an incremental learning scheme as in [9]. A disadvantage of that approach is that only one node is carrying out computations at any time step, whereas in Newscast EM all nodes are running the same protocol in parallel. This results in a batch M-step that has average runtime at most logarithmic in the number of nodes, as we will see next.

## 2  Gaussian mixtures and the EM algorithm

A $k$-component Gaussian mixture for a random vector $x \in \mathbb{R}^d$ is defined as the convex combination

$$p(x) = \sum_{s=1}^{k} \pi_s p(x|s) \tag{1}$$

of $k$ Gaussian densities $p(x|s) = (2\pi)^{-d/2}|C_s|^{-1/2} \exp[-(x - m_s)^\top C_s^{-1}(x - m_s)/2]$, each parameterized by its mean $m_s$ and covariance matrix $C_s$. The components of the mixture are indexed by the random variable $s$ that takes values from 1 to $k$, and $\pi_s = p(s)$ defines a discrete prior distribution over the components. Given a set $\{x_1, \ldots, x_n\}$ of independent and identically distributed samples from $p(x)$, the learning task is to estimate the parameter vector $\theta = \{\pi_s, m_s, C_s\}_{s=1}^k$ of the $k$ components that maximizes the log-likelihood $\mathcal{L} = \sum_{i=1}^{n} \log p(x_i; \theta)$. Throughout we assume that the likelihood function is bounded from above (e.g., by placing appropriate bounds on the components covariance matrices).

Maximization of the data log-likelihood $\mathcal{L}$ can be carried out by the EM algorithm [6] which can be seen as iteratively maximizing a lower bound of $\mathcal{L}$ [9]. This bound $\mathcal{F}$ is a function of the current mixture parameters $\theta$ and a set of 'responsibility' distributions $\{q_i(s)\}$, $i = 1, \ldots, n$, where each $q_i(s)$ corresponds to a data point $x_i$ and defines an arbitrary discrete distribution over $s$. This lower bound is given by:

$$\mathcal{F} = \sum_{i=1}^{n} \sum_{s=1}^{k} q_i(s) \big[ \log p(x_i, s; \theta) - \log q_i(s) \big]. \tag{2}$$

In the E-step of the EM algorithm, the responsibility $q_i(s)$ for each point $x_i$ is set to the Bayes posterior $p(s|x_i)$ given the parameters found in the previous step. In the M-step we solve for the unknown parameters of the mixture by maximizing $\mathcal{F}$ for fixed $q_i(s)$. This yields the following updates:

$$\pi_s = \frac{\sum_{i=1}^{n} q_i(s)}{n}, \qquad m_s = \frac{\sum_{i=1}^{n} q_i(s)x_i}{n\pi_s}, \qquad C_s = \frac{\sum_{i=1}^{n} q_i(s)x_i x_i^\top}{n\pi_s} - m_s m_s^\top. \tag{3}$$

Note that the main operation of the M-step is *averaging*: $\pi_s$ is the average of $q_i(s)$, $m_s$ is the average of products $q_i(s)x_i$ (divided by $\pi_s$), and the covariance matrix $C_s$ is the average of matrices $q_i(s)x_i x_i^\top$ (divided by $\pi_s$ and decreased by $m_s m_s^\top$). This observation is essential for the proposed algorithm, as we will shortly see.

## 3 Newscast computing and averaging

The proposed distributed EM algorithm for Gaussian mixture learning relies on the use of the *Newscast* protocol for distributed computing [3]. Newscast is a gossip-based protocol that applies to networks where arbitrary point-to-point communication between nodes is possible, and it involves repeated data exchange between nodes using randomization: with constant frequency each node contacts some other node at random, and the two nodes exchange application-specific data as well as caches with addresses of other nodes. The protocol is very robust, scalable, and simple to implement—its Java implementation is only a few kBytes of code and can run on small network-enabled computing devices such as mobile phones, PDAs, or sensors.

As with other gossip-based protocols [2], Newscast can be used for computing the mean of a set of values that are distributed over a network. Suppose that values $v_1, \ldots, v_n$ are stored in the nodes of a network, one value per node. Moreover suppose that each node knows the addresses of all other nodes. To compute $\mu = \frac{1}{n} \sum_{i=1}^{n} v_i$, each node $i$ initially sets $\mu_i = v_i$ as its local estimate of $\mu$, and then runs the following protocol for a number of cycles:

---
**Uniform Newscast** (for node $i$)

---
1. Contact a node $j = f(i)$ that is chosen uniformly at random from $1, \ldots, n$.
2. Nodes $i$ and $j$ update their estimates as follows: $\mu_i' = \mu_j' = (\mu_i + \mu_j)/2$.

---

For the purpose of analysis we will assume that in each cycle every node initiates a single contact (but in practice the algorithm can be fully asynchronous). Note that the mean of the local estimates $\{\mu_i\}$ is always the correct mean $\mu$, while for their variance holds:

**Lemma 1.** *In each cycle of uniform Newscast the variance of the local estimates drops on the average by factor $\lambda$, with $\lambda \leq \frac{1}{2\sqrt{e}}$.*

*Proof.*[1] Let $\Phi_t = \sum_{i=1}^{n} (\mu_i - \mu)^2$ be the unnormalized variance of the local estimates $\mu_i$ at cycle $t$. Suppose, without loss of generality, that within cycle $t$ nodes initiate contacts in the order $1, 2, \ldots, n$. The new variance after node's 1 contact is:

$$\Phi_1 = \Phi_t - (\mu_1 - \mu)^2 - (\mu_{f(1)} - \mu)^2 + 2\left(\frac{\mu_1 + \mu_{f(1)}}{2} - \mu\right)^2 \tag{4}$$

$$= \Phi_t - \frac{1}{2}(\mu_1 - \mu)^2 - \frac{1}{2}(\mu_{f(1)} - \mu)^2 + (\mu_1 - \mu)(\mu_{f(1)} - \mu). \tag{5}$$

Taking expectation over $f$, and using the fact that $P[f(i) = j] = \frac{1}{n}$ for all $i$ and $j$, gives:

$$E[\Phi_1 | \Phi_t = \phi] = \phi - \frac{1}{2}(\mu_1 - \mu)^2 - \frac{1}{2n}\sum_{j=1}^{n}(\mu_j - \mu)^2 = \left(1 - \frac{1}{2n}\right)\phi - \frac{1}{2}(\mu_1 - \mu)^2. \tag{6}$$

After $n$ such exchanges, the variance $\Phi_{t+1}$ is on the average:

$$E[\Phi_{t+1} | \Phi_t = \phi] = \left(1 - \frac{1}{2n}\right)^n \phi - \frac{1}{2}\sum_{i=1}^{n}\left(1 - \frac{1}{2n}\right)^{n-i}(\mu_i - \mu)^2. \tag{7}$$

Bounding the term $(1 - \frac{1}{2n})^{n-i}$ by $(1 - \frac{1}{2n})^n$ finally gives:

$$E[\Phi_{t+1} | \Phi_t = \phi] \leq \frac{1}{2}\left(1 - \frac{1}{2n}\right)^n \phi \leq \frac{\phi}{2\sqrt{e}}. \tag{8}$$

$\square$

Thus after $t$ cycles of uniform Newscast, the original variance $\phi_0$ of the local estimates is reduced on the average to $\phi_t \leq \phi_0/(2\sqrt{e})^t$. The fact that the variance drops at an exponential rate means that the nodes learn the correct average very fast. Indeed, using Chebyshev's inequality $P_t[|\mu_i - \mu| \geq \varepsilon] \leq \phi_t/(n\varepsilon^2)$ we see that for any $\varepsilon > 0$, the probability that some node makes an estimation error larger than $\varepsilon$ is dropping exponentially fast with the number of cycles $t$. In particular, we can derive a bound on the number of cycles that are needed in order to guarantee with high probability that *all* nodes know the correct answer with some specific accuracy:

**Theorem 1.** *With probability $1 - \delta$, after $\lceil 0.581(\log n + 2\log \sigma + 2\log \frac{1}{\varepsilon} + \log \frac{1}{\delta}) \rceil$ cycles of uniform Newscast holds $\max_i |\mu_i - \mu| \leq \varepsilon$, for any $\varepsilon > 0$ and data variance $\sigma^2$.*

*Proof.* Using Lemma 1 and the fact that $\phi_0 = n\sigma^2$, we obtain $E[\Phi_t] \leq n\sigma^2/(2\sqrt{e})^t$. Setting $\tau = \log(\frac{n\sigma^2}{\varepsilon^2\delta})/\log(2\sqrt{e})$ we obtain $E[\Phi_\tau] \leq \varepsilon^2\delta$. Using Markov inequality, with probability at least $1 - \delta$ holds $\Phi_\tau \leq \varepsilon^2$. Therefore, since $\Phi_\tau$ is the sum of local terms, for each of them must hold $|\mu_i - \mu| \leq \varepsilon$. It is straightforward to show by induction over $\tau$ that the same inequality will hold for any time $\tau' > \tau$. $\square$

For example, for unit-variance data and a network with $n = 10^4$ nodes we need 49 cycles to guarantee with probability 95% that each node is within $10^{-10}$ from the correct answer.

Note that in uniform Newscast, each node in the network is assumed to know the addresses of all other nodes, and therefore can choose in each cycle one node uniformly at random to exchange data with. In practice, however, each node can only have a limited cache of addresses of other nodes. In this case, the averaging algorithm is modified as follows:

---

**Non-uniform Newscast** (for node $i$)

---

1. Contact a node $j = f(i)$ that is appropriately chosen from $i$'s local cache.
2. Nodes $i$ and $j$ update their estimates as follows: $\mu_i' = \mu_j' = (\mu_i + \mu_j)/2$.
3. Nodes $i$ and $j$ update their caches appropriately.

---

Step 3 implements a 'membership management' schedule which effectively defines a dynamically changing random graph topology over the network. In our experiments we adopted the protocol of [10] which roughly operates as follows. Each entry $k$ in node's $i$ cache contains an 'age' attribute that indicates the number of cycles that have been elapsed since node $k$ created that entry. In step 1 above, node $i$ contacts the node $j$ with the largest age from $i$'s cache, and increases by one the age of all other entries in $i$'s cache. Then node $i$ exchanges estimates with node $j$ as in step 2. In step 3, both nodes $i$ and $j$ select a random subset of their cache entries and mutually exchange them, filling empty slots and discarding self-pointers and duplicates. Finally node $i$ creates an entry with $i$'s address in it and age zero, which is added in $j$'s cache. The resulting protocol is particularly effective and, as we show in the experiments below, in some cases it even outperforms the uniform Newscast. We refer to [10] for more details.

## 4   The Newscast EM algorithm

Newscast EM (NEM) is a gossip-based distributed implementation of the EM algorithm for Gaussian mixture learning, that applies to the following setting. We are given a set of data $\{x_i\}$ that are distributed over the nodes of a network (one data point per node). The data are assumed independent samples from a common $k$-component Gaussian mixture $p(x)$ with (unknown) parameters $\theta = \{\pi_s, m_s, C_s\}_{s=1}^k$. The task is to learn the parameters of the mixture with maximum likelihood in a *decentralized* manner: that is, all learning steps

should be performed locally at the nodes, and they should involve as little communication as possible.

The NEM algorithm is a direct application of the averaging protocol of Section 3 for estimating the parameters $\theta$ of $p(x)$ using the EM updates (3). The E-step of NEM is identical to the E-step of the standard EM algorithm, and it can be performed by all nodes in parallel. The novel characteristic of NEM is the M-step which is implemented as a sequence of gossip-based cycles: At the beginning of each M-step, each node $i$ starts with a local estimate $\theta_i$ of the 'correct' parameter vector $\theta$ (correct according to EM and for the current EM iteration). Then, in every cycle, each node contacts some other node at random, and the two nodes replace their local estimates $\theta_i$ by their (weighted) averages. At the end of the M-step each node has converged (within machine precision) to the correct parameter $\theta$.

To simplify notation, we will denote by $\theta_i = \{\pi_{si}, m_{si}, \tilde{C}_{si}\}$ the local estimates of node $i$ for the parameters of component $s$ at any point of the algorithm. The parameter $\tilde{C}_{si}$ is defined such that $C_{si} = \tilde{C}_{si} - m_{si}m_{si}^\top$. The complete algorithm, which runs identically and in parallel for each node, is as follows:

---

**Newscast EM** (for node $i$)

---

1. **Initialization.** Set $q_i(s)$ to some random positive value and then normalize all $q_i(s)$ to sum to 1 over all $s$.

2. **M-step.** Initialize $i$'s local estimates for each component $s$ as follows: $\pi_{si} = q_i(s)$, $m_{si} = x_i$, $\tilde{C}_{si} = x_i x_i^\top$. Then repeat for $\tau$ cycles:
   a. Contact a node $j = f(i)$ from $i$'s local cache.
   b. Nodes $i$ and $j$ update their local estimates for each component $s$ as follows:

$$\pi'_{si} = \pi'_{sj} = \frac{\pi_{si} + \pi_{sj}}{2}, \tag{9}$$

$$m'_{si} = m'_{sj} = \frac{\pi_{si}m_{si} + \pi_{sj}m_{sj}}{\pi_{si} + \pi_{sj}}, \tag{10}$$

$$\tilde{C}'_{si} = \tilde{C}'_{sj} = \frac{\pi_{si}\tilde{C}_{si} + \pi_{sj}\tilde{C}_{sj}}{\pi_{si} + \pi_{sj}}. \tag{11}$$

   c. Nodes $i$ and $j$ update their caches appropriately.

3. **E-step.** Compute new responsibilities $q_i(s) = p(s|x_i)$ for each component $s$ using the M-step estimates $\pi_{si}$, $m_{si}$, and $C_{si} = \tilde{C}_{si} - m_{si}m_{si}^\top$.

4. **Loop.** Go to step 2, unless a stopping criterion is satisfied that involves the parameter estimates themselves or the energy $\mathcal{F}$.[2]

---

A few observations about the algorithm are in order. First note that both the initialization of the algorithm (step 1) as well as the E-step are completely local to each node. Similarly, a stopping criterion involving the parameter estimates can be implemented locally if each node caches its estimates from the previous EM-iteration. The M-step involves a total of $k[1 + d + d(d + 1)/2]$ averages, for each one of the $k$ components and for dimensionality $d$, which are computed with the Newscast protocol. Given that all nodes agree on the number $\tau$ of Newscast cycles in the M-step, and assuming that $\tau$ is large enough to guarantee convergence to the correct parameter estimates, the complete NEM algorithm can be performed identically and in parallel by all nodes in the network.

It is easy to see that at any cycle of an M-step, and for any component $s$, the weighted

averages over all nodes of the local estimates are always the EM-correct estimates, i.e.,

$$\frac{\sum_{i=1}^{n} \pi_{si} m_{si}}{\sum_{i=1}^{n} \pi_{si}} = m_s \tag{12}$$

and similarly for the $\tilde{C}_{si}$. Moreover, note that the weighted averages of the $m_{si}$ in (10) and the $\tilde{C}_{si}$ in (11), with weights given by (9), can be written as unweighted averages of the corresponding products $\pi_{si} m_{si}$ and $\pi_{si} \tilde{C}_{si}$. In other words, each local estimate can be written as the ratio of two local estimates that converge to the correct values at the same exponential rate (as shown in the previous section). The above observations establish the following:

**Theorem 2.** *In every M-step of Newscast EM, each node converges exponentially fast to the correct parameter estimates for each component of the mixture.*

Similarly, the number of cycles $\tau$ for each M-step can be chosen according to Theorem 1. However, note that in every M-step each node has to wait $\tau$ cycles before its local estimates have converged, and only then can it use these estimates in a new next E-step. We describe here a modification of NEM that allows a node to run a local E-step before its M-step has converged. This 'partial' NEM algorithm is based on the following 'self-correction' idea: instead of waiting until the M-step converges, after a small number of cycles each node runs a local E-step, adjusts its responsibilities, and propagates appropriate corrections through the network.

Such a scheme additionally requires that each node caches its responsibilities from the previous E-step, denoted by $\tilde{q}_i(s)$. The only modification is in the initialization of the M-step: instead of fully resetting the local estimates as in step 2 above, a node makes the following corrections to its current estimates $\pi_{si}, m_{si}, \tilde{C}_{si}$ for each component $s$:

$$\pi'_{si} = \pi_{si} + q_i(s) - \tilde{q}_i(s), \tag{13}$$

$$m'_{si} = \{m_{si} \pi_{si} + x_i[q_i(s) - \tilde{q}_i(s)]\}/\pi'_{si}, \tag{14}$$

$$\tilde{C}'_{si} = \{\tilde{C}_{si} \pi_{si} + x_i x_i^{\top}[q_i(s) - \tilde{q}_i(s)]\}/\pi'_{si}. \tag{15}$$

After these corrections, the Newscast averaging protocol is executed for a number of cycles (smaller than the number $\tau$ of the 'full' NEM). These corrections may increase the variance of the local estimates, but in most cases the corresponding increase of variance is relatively small. This results in speed-ups (often as large as 10-fold), however guaranteed convergence is hard to establish.[3]

## 5   Experiments

To get an insight into the behavior of the presented algorithms we ran several experiments using a Newscast simulator.[4] In Fig. 1 we demonstrate the the performance of uniform and non-uniform Newscast in typical averaging tasks involving zero-mean unit-variance data. In Fig. 1(left) we plot the variance reduction rate $\lambda$ (mean and one standard deviation for 50 runs) as a function of the number of cycles, for averaging problems involving $n = 10^5$ data. Note that in uniform Newscast the observed rate is always below the derived bound $1/(2\sqrt{e}) \approx 0.303$ from Lemma 1. Moreover note that in non-uniform Newscast the variance drops faster than in uniform Newscast. This is due to the fact that the dynamic cache exchange scheme of [10] results in *in-degree* network distributions that are very peaked around the cache size. In practice this means that on the average each node is

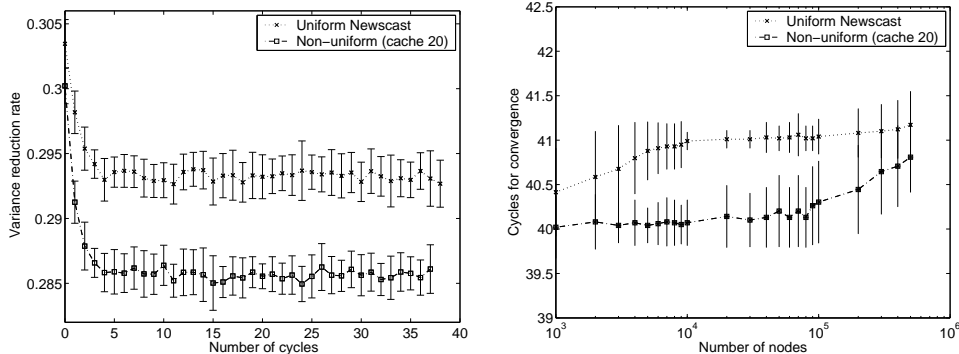

Figure 1: (Left) Variance reduction rate of uniform and non-uniform Newscast, in averaging tasks involving $n = 10^5$ nodes. (Right) Number of cycles to achieve convergence within $\varepsilon = 10^{-10}$ for unit-variance datasets of various sizes.

equally often contacted to by other nodes in each cycle of the protocol. We also observed that the variance reduction rate is on the average unaffected by the network size, while larger networks result in smaller deviations. For $n = 8 * 10^5$, for instance, the standard deviation is half the one plotted above.

In Fig. 1(right) we plot the number of cycles that are required to achieve model accuracy at all nodes within $\varepsilon = 10^{-10}$ as a function of the network size. Note that all observed quantities are below the derived bound of Theorem 1, while non-uniform Newscast performs slightly better than uniform Newscast.

We also ran experiments involving synthetic data drawn from Gaussian mixtures of different number of data points, where we observed results essentially identical to those obtained by the standard (centralized) EM. We also performed some experiments with the 'partial' NEM, where it turned out that in most cases we could obtain the same model accuracy with a much smaller number of cycles (5–10 times than the 'full' NEM), but in some cases the algorithm did not converge.

## 6   Summary and extensions

We presented Newscast EM, a distributed gossip-based implementation of the EM algorithm for learning Gaussian mixture models. Newscast EM applies on networks where each one of a (large) number of nodes observes a local quantity, and can communicate with other nodes in a point-to-point fashion. The algorithm utilizes a gossip-based protocol in its M-step to learn a global Gaussian mixture model from the data: each node starts with a local estimate of the parameters of the mixture and then, for a number of cycles till convergence, pairs of nodes repeatedly exchange their local parameter estimates and combine them by (weighted) averaging. Newscast EM implements a batch M-step that has average runtime logarithmic in the network size. We believe that gossip-based protocols like Newscast can be used in several other algorithms that learn models from distributed data.

Several extensions of the algorithm are possible. Here we have assumed that each node in the network observes one data point. We can easily generalize this to situations where each node observes (and stores) a collection of points, like in [8]. On the other hand, if the locally observed data are too many, one may consider storing only some sufficient statistics of these data locally, and appropriately bound the energy $\mathcal{F}$ in each iteration to get a convergent EM algorithm [11]. Another interesting extension is to replace the averaging

step 2 of uniform and non-uniform Newscast with weighted averaging (for some choice of weights), and study the variance reduction rate in this case. Another interesting problem is when the E-step cannot be performed locally at a node but it requires distributing some information over the network. This could be the case, for instance, when each node observes only a few elements of a vector-valued quantity while, for instance, all nodes together observe the complete sample. We note that if the component models factorize, several useful quantities can be computed by averaging in the log domain. Finally, it would be interesting to investigate the applicability of the Newscast protocol in problems involving distributed inference/learning in more general graphical models [12].

## Acknowledgments

We want to thank Y. Sfakianakis for helping in the experiments, T. Pylak for making his Newscast simulator publicly available, and D. Barber, Z. Ghahramani, and J.J. Verbeek for their comments. N. Vlassis is supported by PROGRESS, the embedded systems research program of the Dutch organization for Scientific Research NWO, the Dutch Ministry of Economic Affairs and the Technology Foundation STW, project AES 5414.

## Footnotes

[1] See [3] for an alternative proof of the same bound.

[2]Note that $\mathcal{F}$ is a sum of local terms, and thus it can also be computed using the same protocol.

[3]This would require, for instance, that individual nodes have estimates of the total variance over the network, which is not obvious how it can be done.

[4]Available from `http://www.cs.vu.nl/~steen/globesoul/sim.tgz`

## References

[1] R. Karp, C. Schindelhauer, S. Shenker, and B. Vöcking. Randomized rumour spreading. In *Proc. 41th IEEE Symp. on Foundations of Computer Science*, Redondo Beach, CA, November 2000.

[2] D. Kempe, A. Dobra, and J. Gehrke. Gossip-based computation of aggregate information. In *Proc. 44th IEEE Symp. on Foundations of Computer Science*, Cambridge, MA, October 2003.

[3] M. Jelasity, W. Kowalczyk, and M. van Steen. Newscast computing. Technical report, Dept. of Computer Science, Vrije Universiteit Amsterdam, 2003. IR-CS-006.

[4] C. C. Moallemi and B. Van Roy. Distributed optimization in adaptive networks. In S. Thrun, L. Saul, and B. Schölkopf, editors, *Advances in Neural Information Processing Systems 16*. MIT Press, Cambridge, MA, 2004.

[5] D. Kempe and F. McSherry. A decentralized algorithm for spectral analysis. In *Proc. 36th ACM Symp. on Theory of Computing*, Chicago, IL, June 2004.

[6] A. P. Dempster, N. M. Laird, and D. B. Rubin. Maximum likelihood from incomplete data via the EM algorithm. *J. Roy. Statist. Soc. B*, 39:1–38, 1977.

[7] G. Forman and B. Zhang. Distributed data clustering can be effi cient and exact. *ACM SIGKDD Explorations*, 2(2):34–38, 2000.

[8] R. D. Nowak. Distributed EM algorithms for density estimation and clustering in sensor networks. *IEEE Trans. on Signal Processing*, 51(8):2245–2253, August 2003.

[9] R. M. Neal and G. E. Hinton. A view of the EM algorithm that justifies incremental, sparse, and other variants. In M. I. Jordan, editor, *Learning in graphical models*, pages 355–368. Kluwer Academic Publishers, 1998.

[10] S. Voulgaris, D. Gavidia, and M. van Steen. Inexpensive membership management for unstructured P2P overlays. *Journal of Network and Systems Management*, 2005. To appear.

[11] J. R. J. Nunnink, J. J. Verbeek, and N. Vlassis. Accelerated greedy mixture learning. In *Proc. Belgian-Dutch Conference on Machine Learning*, Brussels, Belgium, January 2004.

[12] M. A. Paskin and C. E. Guestrin. Robust probabilistic inference in distributed systems. In *Proc. 20th Int. Conf. on Uncertainty in Artificial Intelligence*, Banff, Canada, July 2004.
